# Generalization error bounds for classifiers trained with interdependent data

**Nicolas Usunier, Massih-Reza Amini, Patrick Gallinari**
Department of Computer Science, University of Paris VI
8, rue du Capitaine Scott, 75015 Paris France
{usunier, amini, gallinari}@poleia.lip6.fr

## Abstract

In this paper we propose a general framework to study the generalization properties of binary classifiers trained with data which may be dependent, but are deterministically generated upon a sample of independent examples. It provides generalization bounds for binary classification and some cases of ranking problems, and clarifies the relationship between these learning tasks.

## 1  Introduction

Many machine learning (ML) applications deal with the problem of *bipartite ranking* where the goal is to find a function which orders relevant elements over irrelevant ones. Such problems appear for example in Information Retrieval, where the system returns a list of documents, ordered by relevancy to the user's demand. The criterion widely used to measure the ranking quality is the Area Under the ROC Curve (AUC) [6]. Given a training set $S = ((x_p, y_p))_{p=1}^n$ with $y_p \in \{\pm 1\}$, its optimization over a class of real valued functions $\mathcal{G}$ can be carried out by finding a classifier of the form $c_g(x, x') = sign(g(x) - g(x')), g \in \mathcal{G}$ which minimizes the error rate over pairs of examples $(x, 1)$ and $(x', -1)$ in $S$ [6]. More generally, it is well-known that the learning of scoring functions can be expressed as a classification task over pairs of examples [7, 5].

The study of the generalization properties of ranking problems is a challenging task, since the pairs of examples violate the central i.i.d. assumption of binary classification. Using task-specific studies, this issue has recently been the focus of a large amount of work. [2] showed that SVM-like algorithms optimizing the AUC have good generalization guarantees, and [11] showed that maximizing the margin of the pairs, defined by the quantity $g(x) - g(x')$, leads to the minimization of the generalization error. While these results suggest some similarity between the classification of the pairs of examples and the classification of independent data, no common framework has been established. As a major drawback, it is not possible to directly deduce results for ranking from those obtained in classification.

In this paper, we present a new framework to study the generalization properties of classifiers over data which can exhibit a suitable dependency structure. Among others, the problems of binary classification, bipartite ranking, and the *ranking risk* defined in [5] are special cases of our study. It shows that it is possible to infer generalization bounds for clas-

sifiers trained over interdependent examples using generalization results known for binary classification. We illustrate this property by proving a new margin-based, data-dependent bound for SVM-like algorithms optimizing the AUC. This bound derives straightforwardly from the same kind of bounds for SVMs for classification given in [12]. Since learning algorithms aim at minimizing the generalization error of their chosen hypothesis, our results suggest that the design of bipartite ranking algorithms can follow the design of standard classification learning systems.

The remainder of this paper is as follows. In section 2, we give the formal definition of our framework and detail the progression of our analysis over the paper. In section 3, we present a new concentration inequality which allows to extend the notion of Rademacher complexity (section 4), and, in section 5, we prove generalization bounds for binary classification and bipartite ranking tasks under our framework. Finally, the missing proofs are given in a longer version of the paper [13].

## 2 Formal framework

We distinguish between the *input* and the *training* data. The input data $S = (s_p)_{p=1}^n$ is a set of $n$ independent examples, while the training data $Z = (z_i)_{i=1}^N$ is composed of $N$ binary classified elements where each $z_i$ is in $\mathcal{X}_{tr} \times \{-1, +1\}$, with $\mathcal{X}_{tr}$ the space of characteristics. For example, in the general case of bipartite ranking, the *input data* is the set of elements to be ordered, while the *training data* is constituted by the pairs of examples to be classified. The purpose of this work is the study of generalization properties of classifiers trained using a possibly dependent *training data*, but in the special case where the latter is deterministically generated from the *input data*. The aim here is to select a hypothesis $h \in \mathcal{H} = \{h_\theta : \mathcal{X}_{tr} \to \{-1, 1\} | \theta \in \Theta\}$ which optimizes the empirical risk $L(h, Z) = \frac{1}{N} \sum_{i=1}^N \ell(h, z_i)$, $\ell$ being the instantaneous loss of $h$, over the training set $Z$.

**Definition 1** (Classifiers trained with interdependent data). *A classification algorithm over interdependent training data takes as input data a set $S = (s_p)_{p=1}^n$ supposed to be drawn according to an unknown product distribution $\otimes_{p=1}^n \mathcal{D}_p$ over a product sample space $\mathcal{S}^n$ [1], outputs a binary classifier chosen in a hypothesis space $\mathcal{H} : \{h : \mathcal{X}_{tr} \to \{+1, -1\}\}$, and has a two-step learning process. In a first step, the learner applies to its input data $S$ a fixed function $\varphi : \mathcal{S}^n \to (\mathcal{X}_{tr} \times \{-1, 1\})^N$ to generate a vector $Z = (z_i)_{i=1}^N = \varphi(S)$ of $N$ training examples $z_i \in \mathcal{X}_{tr} \times \{-1, 1\}, i = 1, ..., N$. In the second step, the learner runs a classification algorithm in order to obtain $h$ which minimizes the empirical classification loss $L(h, Z)$, over its training data $Z = \varphi(S)$.*

**Examples** Using the notations above, when $\mathcal{S} = \mathcal{X}_{tr} \times \{\pm 1\}$, $n = N$, $\varphi$ is the identity function and $S$ is drawn i.i.d. according to an unknown distribution $\mathcal{D}$, we recover the classical definition of a binary classification algorithm. Another example is the ranking task described in [5] where $\mathcal{S} = \mathcal{X} \times \mathbb{R}$, $\mathcal{X}_{tr} = \mathcal{X}^2$, $N = n(n-1)$ and, given $S = ((x_p, y_p))_{p=1}^n$ drawn i.i.d. according to a fixed $\mathcal{D}$, $\varphi$ generates all the pairs $((x_k, x_l), sign(\frac{y_k - y_l}{2}))$, $k \neq l$.

In the remaining of the paper, we will prove generalization error bounds of the selected hypothesis by upper bounding

$$\sup_{h \in \mathcal{H}} L(h) - L(h, \varphi(S)) \tag{1}$$

with high confidence over $S$, where $L(h) = \mathbb{E}_S L(h, \varphi(S))$. To this end we decompose $Z = \varphi(S)$ using the dependency graph of the random variables composing $Z$ with a technique similar to the one proposed by [8]. We go towards this result by first bounding

$\sup_{q \in \mathcal{Q}} \left[ \mathbb{E}_{\tilde{S}} \frac{1}{N} \sum_{i=1}^{N} q(\varphi(\tilde{S})_i) - \frac{1}{N} \sum_{i=1}^{N} q(\varphi(S)_i) \right]$ with high confidence over samples $S$, where $\tilde{S}$ is also drawn according to $\otimes_{p=1}^{n} \mathcal{D}_p$, $\mathcal{Q}$ is a class of functions taking values in $[0, 1]$, and $\varphi(S)_i$ denotes the $i$-th training example (Theorem 4). This bound uses an extension of the Rademacher complexity [3], the fractional Rademacher complexity (FRC) (definition 3), which is a weighted sum of Rademacher complexities over independent subsets of the training data. We show that the FRC of an arbitrary class of real-valued functions can be trivially computed given the Rademacher complexity of this class of functions and $\varphi$ (theorem 6). This theorem shows that generalization error bounds for classes of classifiers over interdependent data (in the sense of definition 1) trivially follows from the same kind of bounds for the same class of classifiers trained over i.i.d. data. Finally, we show an example of the derivation of a margin-based, data-dependent generalization error bound (i.e. a bound on equation (1) which can be computed on the training data) for the bipartite ranking case when $\mathcal{H} = \{(x, x') \mapsto sign(K(\theta, x) - K(\theta, x')) | K(\theta, \theta) \le B^2\}$, assuming that the input examples are drawn i.i.d. according to a distribution $\mathcal{D}$ over $\mathcal{X} \times \{\pm 1\}$, $\mathcal{X} \subset \mathbb{R}^d$ and $K$ is a kernel over $\mathcal{X}^2$.

**Notations** Throughout the paper, we will use the notations of the preceding subsection, except for $Z = (z_i)_{i=1}^{N}$, which will denote an arbitrary element of $(\mathcal{X}_{tr} \times \{-1, 1\})^N$. In order to obtain the dependency graph of the random variables $\varphi(S)_i$, we will consider, for each $1 \le i \le N$, a set $[i] \subset \{1, ..., n\}$ such that $\varphi(S)_i$ depends only on the variables $s_p \in S$ for which $p \in [i]$. Using these notations, if we consider two indices $k, l$ in $\{1, ..., N\}$, we can notice that the two random variables $\varphi(S)_k$ and $\varphi(S)_l$ are independent if and only if $[k] \cap [l] = \emptyset$. The dependency graph of the $\varphi(S)_i$s follows, by constructing the graph $\Gamma(\varphi)$, with the set of vertices $V = \{1, ..., N\}$, and with an edge between $k$ and $l$ if and only if $[k] \cap [l] \ne \emptyset$. The following definitions, taken from [8], will enable us to separate the set of partly dependent variables into sets of independent variables:

- A subset $A$ of $V$ is independent if all the elements in $A$ are independent.
- A sequence $\mathcal{C} = (C_j)_{j=1}^{m}$ of subsets of $V$ is a proper cover of $V$ if, for all $j$, $C_j$ is independent, and $\bigcup_j C_j = V$
- A sequence $\mathcal{C} = (C_j, w_j)_{j=1}^{m}$ is a proper, exact fractional cover of $\Gamma$ if $w_j > 0$ for all $j$, and, for each $i \in V$, $\sum_{j=1}^{m} w_j \mathbf{I}_{C_j}(i) = 1$, where $\mathbf{I}_{C_j}$ is the indicator function of $C_j$.
- The fractional chromatic number of $\Gamma$, noted $\chi(\Gamma)$, is equal to the minimum of $\sum_j w_j$ over all proper, exact fractional cover.

It is to be noted that from lemma 3.2 of [8], the existence of proper, exact fractional covers is ensured. Since $\Gamma$ is fully determined by the function $\varphi$, we will note $\chi(\Gamma) = \chi(\varphi)$. Moreover, we will denote by $\mathcal{C}(\varphi) = (C_j, w_j)_{j=1}^{\kappa}$ a proper, exact fractional cover of $\Gamma$ such that $\sum_j w_j = \chi(\varphi)$. Finally, for a given $\mathcal{C}(\varphi)$, we denote by $\kappa_j$ the number of elements in $C_j$, and we fix the notations: $C_j = \{C_{j1}, ..., C_{j\kappa_j}\}$. It is to be noted that if $(t_i)_{i=1}^{N} \in \mathbb{R}^N$, and $\mathcal{C}(\varphi) = (C_j, w_j)_{j=1}^{\kappa}$, lemma 3.1 of [8] states that:

$$\sum_{i=1}^{N} t_i = \sum_{j=1}^{\kappa} w_j T_j, \text{where } T_j = \sum_{k=1}^{\kappa_j} t_{C_j k} \tag{2}$$

## 3  A new concentration inequality

Concentration inequalities bound the probability that a random variable deviates *too much* from its expectation (see [4] for a survey). They play a major role in learning theory as

they can be used for example to bound the probability of deviation of the expected loss of a function from its empirical value estimated over a sample set. A well-known inequality is McDiarmid's theorem [9] for independent random variables, which bounds the probability of deviation from its expectation of an arbitrary function with bounded variations over each one of its parameters. While this theorem is very general, [8] proved a large deviation bound for sums of partly random variables where the dependency structure of the variables is known, which can be tighter in some cases. Since we also consider variables with known dependency structure, using such results may lead to tighter bounds. However, we will bound functions like in equation (1), which do not write as a sum of partly dependent variables. Thus, we need a result on more general functions than sums of random variables, but which also takes into account the known dependency structure of the variables.

**Theorem 2.** *Let* $\varphi : \mathcal{X}^n \to \mathcal{X'}^N$. *Using the notations defined above, let* $\mathcal{C}(\varphi) = (C_j, w_j)_{j=1}^{\kappa}$. *Let* $f : \mathcal{X'}^N \to \mathbb{R}$ *such that:*

1. *There exist* $\kappa$ *functions* $f_j : \mathcal{X'}^{\kappa_j} \to \mathbb{R}$ *which satisfy* $\forall Z = (z_1, ..., z_N) \in \mathcal{X'}^N$, $f(Z) = \sum_j w_j f_j(z_{C_{j1}}, ..., z_{C_{j\kappa_j}})$.

2. *There exist* $\beta_1, ..., \beta_N \in \mathbb{R}_+$ *such that* $\forall j, \forall Z_j, Z_j^k \in \mathcal{X'}^{\kappa_j}$ *such that* $Z_j$ *and* $Z_j^k$ *differ only in the k-th dimension,* $|f_j(Z_j) - f_j(Z_j^k)| \leq \beta_{C_{jk}}$.

*Let finally* $\mathcal{D}_1, ..., \mathcal{D}_n$ *be* $n$ *probability distributions over* $\mathcal{X}$. *Then, we have:*

$$\mathbb{P}_{X \sim \otimes_{i=1}^n \mathcal{D}_i}(f \circ \varphi(X) - \mathbb{E}f \circ \varphi > \epsilon) \leq \exp(-\frac{2\epsilon^2}{\chi(\varphi) \sum_{i=1}^N \beta_i^2}) \tag{3}$$

*and the same holds for* $\mathbb{P}(\mathbb{E}f \circ \varphi - f \circ \varphi > \epsilon)$.

The proof of this theorem (given in [13]) is a variation of the demonstrations in [8] and McDiarmid's theorem. The main idea of this theorem is to allow the decomposition of $f$, which will take as input partly dependent random variables when applied to $\varphi(S)$, into a sum of functions which, when considering $f \circ \varphi(S)$, will be functions of independent variables. As we will see, this theorem will be the major tool in our analysis. It is to be noted that when $\mathcal{X} = \mathcal{X'}$, $N = n$ and $\varphi$ is the identity function of $\mathcal{X}^n$, the theorem 2 is exactly McDiarmid's theorem. On the other hand, when $f$ takes the form $\sum_{i=1}^N q_i(z_i)$ with for all $z \in \mathcal{X'}$, $a \leq q_i(z) \leq a + \beta_i$ with $a \in \mathbb{R}$, then theorem 2 reduces to a particular case of the large deviation bound of [8].

## 4 The fractional Rademacher complexity

Let $Z = (z_i)_{i=1}^N \in \mathcal{Z}^N$. If $Z$ is supposed to be drawn i.i.d. according to a distribution $\mathcal{D}_{\mathcal{Z}}$ over $\mathcal{Z}$, for a class $\mathcal{F}$ of functions from $\mathcal{Z}$ to $\mathbb{R}$, the Rademacher complexity of $\mathcal{F}$ is defined by [10] $R_N(\mathcal{F}) = \mathbb{E}_{Z \sim \mathcal{D}_{\mathcal{Z}}} R_N(\mathcal{F}, Z)$, where $R_N(\mathcal{F}, Z) = \mathbb{E}_{\sigma} \sup_{f \in \mathcal{F}} \sum_{i=1}^N \sigma_i f(z_i)$ is the empirical Rademacher complexity of $\mathcal{F}$ on $Z$, and $\sigma = (\sigma_i)_{i=1}^n$ is a sequence of independent Rademacher variables, i.e. $\forall i, \mathbb{P}(\sigma_i = 1) = \mathbb{P}(\sigma_i = -1) = \frac{1}{2}$. This quantity has been extensively used to measure the complexity of function classes in previous bounds for binary classification [3, 10]. In particular, if we consider a class of functions $\mathcal{Q} = \{q : \mathcal{Z} \mapsto [0, 1]\}$, it can be shown (theorem 4.9 in [12]) that with probability at least $1 - \delta$ over $Z$, all $q \in \mathcal{Q}$ verify the following inequality, which serves as a preliminary result to show data-dependent bounds for SVMs in [12]:

$$\mathbb{E}_{Z \sim \mathcal{D}_{\mathcal{Z}}} q(z) \leq \frac{1}{N} \sum_{i=1}^N q(z_i) + R_N(\mathcal{Q}) + \sqrt{\frac{\ln(1/\delta)}{2N}} \tag{4}$$

In this section, we generalize equation (4) to our case with theorem 4, using the following definition[2] (we denote $\lambda(q, \varphi(S)) = \frac{1}{N} \sum_{i=1}^{N} q(\varphi(S)_i)$ and $\lambda(q) = \mathbb{E}_S \lambda(q, \varphi(S))$):

**Definition 3.** *Let $\mathcal{Q}$, be class of functions from a set $\mathcal{Z}$ to $\mathbb{R}$, Let $\varphi : \mathcal{X}^n \to \mathcal{Z}^N$ and $S$ a sample of size $n$ drawn according to a product distribution $\otimes_{p=1}^n \mathcal{D}_p$ over $\mathcal{X}^n$. Then, we define the empirical fractional Rademacher complexity [3] of $\mathcal{Q}$ given $\varphi$ as:*

$$R_n^*(\mathcal{Q}, S, \varphi) = \frac{2}{N} \mathbb{E}_\sigma \sum_j w_j \sup_{q \in \mathcal{Q}} \sum_{i \in C_j} \sigma_i q(\varphi(S)_i)$$

*As well as the fractional Rademacher complexity of $\mathcal{Q}$ as $R_n^*(\mathcal{Q}, \varphi) = \mathbb{E}_S R_n^*(\mathcal{Q}, S, \varphi)$*

**Theorem 4.** *Let $\mathcal{Q}$ be a class of functions from $\mathcal{Z}$ to $[0, 1]$. Then, with probability at least $1 - \delta$ over the samples $S$ drawn according to $\otimes_{p=1}^n \mathcal{D}_p$, for all $q \in \mathcal{Q}$:*

$$\lambda(q) - \frac{1}{N} \sum_{i=1}^N q(\varphi_i(S)) \leq R_n^*(\mathcal{Q}, \varphi) + \sqrt{\frac{\chi(\varphi) \ln(1/\delta)}{2N}}$$

$$\textit{And: } \quad \lambda(q) - \frac{1}{N} \sum_{i=1}^N q(\varphi_i(S)) \leq R_n^*(\mathcal{Q}, S, \varphi) + 3\sqrt{\frac{\chi(\varphi) \ln(2/\delta)}{2N}}$$

In the definition of the fractional Rademacher complexity (FRC), if $\varphi$ is the identity function, we recover the standard Rademacher complexity, and theorem 4 reduces to equation (4). These results are therefore extensions of equation (4), and show that the generalization error bounds for the tasks falling in our framework will follow from a unique approach.

*Proof.* In order to find a bound for all $q$ in $\mathcal{Q}$ of $\lambda(q) - \lambda(q, \varphi(S))$, we write:

$$\lambda(q) - \lambda(q, \varphi(S)) \leq \sup_{q \in \mathcal{Q}} \left[ \mathbb{E}_{\tilde{S}} \frac{1}{N} \sum_{i=1}^N q(\varphi(\tilde{S})_i) - \frac{1}{N} \sum_{i=1}^N q(\varphi(S)_i) \right]$$

$$\leq \frac{1}{N} \sum_j w_j \sup_{q \in \mathcal{Q}} \left[ \mathbb{E}_{\tilde{S}} \sum_{i \in C_j} q(\varphi(\tilde{S})_i) - \sum_{i \in C_j} q(\varphi(S)_i) \right] \qquad (5)$$

Where we have used equation (2). Now, consider, for each $j$, $f_j : \mathcal{Z}^{\kappa_j} \to \mathbb{R}$ such that, for all $z^{(j)} \in \mathcal{Z}^{\kappa_j}$, $f_j(z^{(j)}) = \frac{1}{N} \sup_{q \in \mathcal{Q}} \mathbb{E}_{\tilde{S}} \sum_{k=1}^{\kappa_j} q(\varphi(\tilde{S})_{C_j k}) - \sum_{k=1}^{\kappa_j} q(z_k^{(j)})$. It is clear that if $f : \mathcal{Z}^N \to \mathbb{R}$ is defined by: for all $Z \in \mathcal{Z}^N$, $f(Z) = \sum_{j=1}^N w_j f_j(z_{Cj1}, ..., z_{Cj\kappa_j})$, then the right side of equation (5) is equal to $f \circ \varphi(S)$, and that $f$ satisfies all the conditions of theorem 2 with, for all $i \in \{1, ..., N\}, \beta_i = \frac{1}{N}$. Therefore, with a direct application of theorem 2, we can claim that, with probability at least $1 - \delta$ over samples $S$ drawn according to $\otimes_{p=1}^n \mathcal{D}_p$ (we denote $\lambda_j(q, \varphi(S)) = \frac{1}{N} \sum_{i \in C_j} q(\varphi(S)_i)$):

$$\lambda(q) - \lambda(q, \varphi(S)) \leq \mathbb{E}_S \sum_j w_j \sup_{q \in \mathcal{Q}} \left[ \mathbb{E}_{\tilde{S}} \lambda_j(q, \varphi(\tilde{S})) - \lambda_j(q, \varphi(S)) \right] + \sqrt{\frac{\chi(\varphi) \ln(1/\delta)}{2N}}$$

$$\leq \mathbb{E}_{S, \tilde{S}} \sum_j \frac{w_j}{N} \sup_{q \in \mathcal{Q}} \sum_{i \in C_j} [q(\varphi(\tilde{S})_i) - q(\varphi(S)_i)] + \sqrt{\frac{\chi(\varphi) \ln(1/\delta)}{2N}}$$

$$(6)$$

Now fix $j$, and consider $\sigma = (\sigma_i)_{i=1}^N$, a sequence of $N$ independent Rademacher variables. For a given realization of $\sigma$, we have that

$$\mathbb{E}_{S,\tilde{S}} \sup_{q\in\mathcal{Q}} \sum_{i\in C_j} [q(\varphi(\tilde{S})_i) - q(\varphi(S)_i)] = \mathbb{E}_{S,\tilde{S}} \sup_{q\in\mathcal{Q}} \sum_{i\in C_j} \sigma_i [q(\varphi(\tilde{S})_i) - q(\varphi(S)_i)] \quad (7)$$

because, for each $\sigma_i$ considered, $\sigma_i = -1$ simply corresponds to permutating, in $S, \tilde{S}$, the two sequences $S_{[i]}$ and $\tilde{S}_{[i]}$ (where $S_{[i]}$ denotes the subset of $S$ $\varphi(S)_i$ really depends on) which have the same distribution (even though the $s_p$'s are not identically distributed), and are independent from the other $S_{[k]}$ and $\tilde{S}_{[k]}$ since we are considering $i, k \in C_j$. Therefore, taking the expection over $S, \tilde{S}$ is the same with the elements permuted this way as if they were not permuted. Then, from equation (6), the first inequality of the theorem follow. The second inequality is due to an application of theorem 2 to $R_n^*(\mathcal{Q}, S, \varphi)$. □

*Remark* 5. The symmetrization performed in equation (7) requires the variables $\varphi(S)_i$ appearing in the same sum to be independent. Thus, the generalization of Rademacher complexities could only be performed using a decomposition in independent sets, and the cover $\mathcal{C}$ assures some optimality of the decomposition. Moreover, even though McDiarmid's theorem could be applied each time we used theorem 2, the derivation of the real numbers bounding the differences is not straightforward, and may not lead to the same result. The creation of the dependency graph of $\varphi$ and theorem 2 are therefore necessary tools for obtaining theorem 4.

**Properties of the fractional Rademacher complexity**

**Theorem 6.** *Let $\mathcal{Q}$ be a class of functions from a set $\mathcal{Z}$ to $\mathbb{R}$, and $\varphi : \mathcal{X}^n \to \mathcal{Z}^N$. For $S \in \mathcal{X}^n$, the following results are true.*

1. *Let $\phi : \mathbb{R} \to \mathbb{R}$, an L-Lipschitz function. Then $R_n^*(\phi \circ \mathcal{Q}, S, \varphi) \leq L R_n^*(\mathcal{Q}, S, \varphi)$*

2. *If there exist $M > 0$ such that for every $k$, and samples $S_k$ of size $k$ $R_k(\mathcal{Q}, S_k) \leq \frac{M}{\sqrt{k}}$, then $R_n^*(\mathcal{Q}, S, \varphi) \leq M\sqrt{\frac{\chi(\varphi)}{N}}$*

3. *Let $K$ be a kernel over $\mathcal{Z}$, $B > 0$, denote $||x||_K = \sqrt{K(x,x)}$ and define $\mathcal{H}_{K,B} = \{h_\theta : \mathcal{Z} \to \mathbb{R}, h_\theta(x) = K(\theta, x)|||\theta||_K \leq B\}$. Then:*

$$R_n^*(\mathcal{H}_{K,B}, S, \varphi) \leq \frac{2B\sqrt{\chi(\varphi)}}{N} \sqrt{\sum_{i=1}^N ||\varphi(S)_i||_K^2}$$

The first point of this theorem is a direct consequence of a Rademacher process comparison theorem, namely theorem 7 of [10], and will enable the obtention of margin-based bounds. The second and third points show that the results regarding the Rademacher complexity can be used to immediately deduce bounds on the FRC. This result, as well as theorem 4 show that binary classifiers of i.i.d. data and classifiers of interdependent data will have generalization bounds of the same form, but with different convergence rate depending on the dependency structure imposed by $\varphi$.

*elements of proof.* The second point results from Jensen's inequality, using the facts that $\sum_j w_j = \chi(\varphi)$ and, from equation (2), $\sum_j w_j |C_j| = N$. The third point is based on the same calculations by noting that (see e.g. [3]), if $S_k = ((x_p, y_p))_{p=1}^k$, then $R_k(\mathcal{H}_{K,B}, S_k) \leq \frac{2B}{k} \sqrt{\sum_{p=1}^k ||x_p||_K^2}$. □

# 5 Data-dependent bounds

The fact that classifiers trained on interdependent data will "inherit" the generalization bound of the same classifier trained on i.i.d. data suggests simple ways of obtaining bipartite ranking algorithms. Indeed, suppose we want to learn a linear ranking function, for example a function $h \in \mathcal{H}_{K,B}$ as defined in theorem 6, where $K$ is a linear kernel, and consider a sample $S \in (\mathcal{X} \times \{-1,1\})^n$ with $\mathcal{X} \subset \mathbb{R}^d$, drawn i.i.d. according to some $\mathcal{D}$. Then we have, for input examples $(x,1)$ and $(x',-1)$ in $S$, $h(x) - h(x') = h(x - x')$. Therefore, we can learn a bipartite ranking function by applying an SVM algorithm to the pairs $((x,1),(x',-1))$ in $S$, each pair being represented by $x - x'$, and our framework allows to immediately obtain generalization bounds for this learning process based on the generalization bounds for SVMs. We show these bounds in theorem 7.

To derive the bounds, we consider $\phi$, the 1-Lipschitz function defined by $\phi(x) = \min(1, \max(1 - x, 0)) \geq [[x \leq 0]]^4$. Given a training example $z$, we denote by $z^l$ its label and $z^f$ its feature representation. With an abuse of notation, we denote $\phi(h, Z) = \frac{1}{N} \sum_{i=1}^{N} \phi(z_i^l h(z_i^f))$. For a sample $S$ drawn according to $\bigotimes_{p=1}^{n} \mathcal{D}_p$, we have, for all $h$ in some function class $\mathcal{H}$:

$$\mathbb{E}_S \frac{1}{N} \sum_i \ell(h, z_i) \leq \mathbb{E}_S \frac{1}{N} \sum_i \phi(z_i^l h(z_i^f))$$

$$\leq \phi(h, Z) + \mathbb{E}_\sigma \sum_j \frac{2w_j}{N} \sup_{h \in \mathcal{H}} \sum_{i \in C_j} \sigma_i \phi(z_i^l h(z_i^f)) + 3\sqrt{\frac{\chi(\varphi) \ln(2/\delta)}{2N}}$$

where $\ell(h, z_i) = [[z_i^l h(z_i^f) \leq 0]]$, and the last inequality holds with probability at least $1 - \delta$ over samples $S$ from theorem 4. Notice that when $\sigma_{C_{jk}}$ is a Rademacher variable, it has the same distribution as $z_{C_{jk}}^l \sigma_{C_{jk}}$ since $z_{C_{jk}}^l \in \{-1, 1\}$. Thus, using the first result of theorem 6 we have that with probability $1 - \delta$ over the samples $S$, all $h$ in $\mathcal{H}$ satisfy:

$$\mathbb{E}_S \frac{1}{N} \sum_i \ell(h, z_i) \leq \frac{1}{N} \sum_i \phi(z^l h(z^f)) + R_n^*(\mathcal{H}, S, \varphi) + 3\sqrt{\frac{\chi(\varphi) \ln(2/\delta)}{2N}} \quad (8)$$

Now putting in equation (8) the third point of theorem 6, with $\mathcal{H} = \mathcal{H}_{K,B}$ as defined in theorem 6 with $\mathcal{Z} = \mathcal{X}$, we obtain the following theorem:

**Theorem 7.** *Let $S \in (\mathcal{X} \times \{-1,1\})^n$ be a sample of size $n$ drawn i.i.d. according to an unknown distribution $\mathcal{D}$. Then, with probability at least $1 - \delta$, all $h \in \mathcal{H}_{K,B}$ verify:*

$$\mathbb{E}_S[[y_i h(x_i) \leq 0]] \leq \frac{1}{n} \sum_{i=1}^{n} \phi(y_i h(x_i)) + \frac{2B}{n}\sqrt{\sum_{i=1}^{n} ||x_i||_K^2} + 3\sqrt{\frac{\ln(2/\delta)}{2n}}$$

*And* $\mathbb{E}\{[[h(x) \leq h(x')]] | y = 1, y' = -1\} \leq \frac{1}{n_+^S n_-^S} \sum_{i=1}^{n_+^S} \sum_{j=1}^{n_-^S} \phi(h(x_{\sigma(i)}) - h(x_{\nu(j)}))$

$$+ \frac{2B\sqrt{\max(n_+^S, n_-^S)}}{n_+^S n_-^S} \sqrt{\sum_{i=1}^{n_+^S} \sum_{j=1}^{n_-^S} ||x_{\sigma(i)} - x_{\nu(j)}||_K^2} + 3\sqrt{\frac{\ln(2/\delta)}{2\min(n_+^S, n_-^S)}}$$

*Where $n_+^S$, $n_-^S$ are the number of positive and negative instances in $S$, and $\sigma$ and $\nu$ also depend on $S$, and are such that $x_{\sigma(i)}$ is the $i$-th positive instance in $S$ and $\nu(j)$ the $j$-th negative instance.*

It is to be noted that when $h \in \mathcal{H}_{K,B}$ with a non linear kernel, the same bounds apply, with, for the case of bipartite ranking, $||x_{\sigma(i)} - x_{\nu(j)}||_K^2$ replaced by $||x_{\sigma(i)}||_K^2 + ||x_{\nu(j)}||_K^2 - 2K(x_{\sigma(i)}, x_{\nu(j)})$.

For binary classification, we recover the bounds of [12], since our framework is a generalization of their approach. As expected, the bounds suggest that kernel machines will generalize well for bipartite ranking. Thus, we recover the results of [2] obtained in a specific framework of algorithmic stability. However, our bound suggests that the convergence rate is controlled by $1/\min(n_+^S, n_-^S)$, while their results suggested $1/n_+^S + 1/n_-^S$. The full proof, in which we follow the approach of [1], is given in [13].

# 6 Conclusion

We have shown a general framework for classifiers trained with interdependent data, and provided the necessary tools to study their generalization properties. It gives a new insight on the close relationship between the binary classification task and the bipartite ranking, and allows to prove the first data-dependent bounds for this latter case. Moreover, the framework could also yield comparable bounds on other learning tasks.

**Acknowledgments**

This work was supported in part by the IST Programme of the European Community, under the PASCAL Network of Excellence, IST-2002-506778. This publication only reflects the authors views.

**References**

[1] Agarwal S., Graepel T., Herbrich R., Har-Peled S., Roth D. (2005) Generalization Error Bounds for the Area Under the ROC curve, *Journal of Machine Learning Research*.

[2] Agarwal S., Niyogi P. (2005) Stability and generalization of bipartite ranking algorithms, *Conference on Learning Theory 18*.

[3] Bartlett P., Mendelson S. (2002) Rademacher and Gaussian Complexities: Risk Bounds and Structural Results, *Journal of Machine Learning Research 3*, pp. 463-482.

[4] Boucheron S., Bousquet O., Lugosi G. (2004) Concentration inequalities, in O. Bousquet, U.v. Luxburg, and G. Rtsch (editors), *Advanced Lectures in Machine Learning*, Springer, pp. 208-240.

[5] Clemençon S., Lugosi G., Vayatis N. (2005) Ranking and scoring using empirical risk minimization, *Conference on Learning Theory 18*.

[6] Cortes C., Mohri M. (2004) AUC optimization vs error rate miniminzation *NIPS 2003*,

[7] Freund Y., Iyer R.D., Schapire R.E., Singer Y. (2003) An Efficient Boosting Algorithm for Combining Preferences, *Journal of Machine Learning Research 4*, pp. 933-969.

[8] Janson S. (2004) Large deviations for sums of partly dependent random variables, *Random Structures and Algorithms 24*, pp. 234-248.

[9] McDiarmid C. (1989) On the method of bounded differences, *Surveys in Combinatorics*.

[10] Meir R., Zhang T. (2003) Generalization Error Bounds for Bayesian Mixture Algorithms, *Journal of Machine Learning Research 4*, pp. 839-860.

[11] Rudin C., Cortes C., Mohri M., Schapire R.E. (2005) Margin-Based Ranking meets Boosting in the middle, *Conference on Learning Theory 18*.

[12] Shawe-Taylor J., Cristianini N. (2004) Kernel Methods for Pattern Analysis, Cambridge U. Prs.

[13] *Long version of this paper*, Available at http://www-connex.lip6.fr/˜usunier/nips05-lv.pdf

## Footnotes

[1]It is equivalent to say that the *input data* is a vector of independent, but not necessarilly identically distributed random variables.

[2]The fractional Rademacher complexity depends on the cover $\mathcal{C}(\varphi)$ chosen, since it is not unique. However in practice, our bounds only depend on $\chi(\varphi)$ (see section 4.1).

[3]this denomination stands as it is a sum of Rademacher averages over independent parts of $\varphi(S)$.

[4] remark that $\phi$ is upper bounded by the slack variables of the SVM optimization problem (see e.g. [12]).
